# Semi-supervised Learning on Directed Graphs

**Dengyong Zhou**[†]**, Bernhard Schölkopf**[†]**, and Thomas Hofmann**[‡†]
[†]Max Planck Institute for Biological Cybernetics
72076 Tuebingen, Germany
{dengyong.zhou, bernhard.schoelkopf}@tuebingen.mpg.de
[‡]Department of Computer Science, Brown University
Providence, RI 02912 USA
th@cs.brown.edu

## Abstract

Given a directed graph in which some of the nodes are labeled, we investigate the question of how to exploit the link structure of the graph to infer the labels of the remaining unlabeled nodes. To that extent we propose a regularization framework for functions defined over nodes of a directed graph that forces the classification function to change slowly on densely linked subgraphs. A powerful, yet computationally simple classification algorithm is derived within the proposed framework. The experimental evaluation on real-world Web classification problems demonstrates encouraging results that validate our approach.

## 1 Introduction

We consider semi-supervised classification problems on weighted directed graphs, in which some nodes in the graph are labeled as positive or negative, and where the task consists in classifying unlabeled nodes. Typical examples of this kind are Web page categorization based on hyperlink structure [4, 11] and document classification or recommendation based on citation graphs [10], yet similar problems exist in other domains such as computational biology. For the sake of concreteness, we will mainly focus on the Web graph in the sequel, i.e. the considered graph represents a subgraph of the Web, where nodes correspond to Web pages and directed edges represent hyperlinks between them (cf. [3]).

We refrain from utilizing attributes or features associated with each node, which may or may not be available in applications, but rather focus on the analysis of the connectivity of the graph as a means for classifying unlabeled nodes. Such an approach inevitably needs to make some *a priori* premise about how connectivity and categorization of individual nodes may be related in real-world graphs. The fundamental assumption of our framework is the *category similarity of co-linked nodes* in a directed graph. This is a slightly more complex concept than in the case of undirected (weighted) graphs [1, 18, 12, 15, 17], where a typical assumption is that an edge connecting two nodes will more or less increase the likelihood of the nodes belonging to the same category. Co-linkage on the other hand seems a more suitable and promising concept in directed graphs, as is witnessed by its successful use in Web page categorization [4] as well as co-citation analysis for information retrieval [10]. Notice that co-linkage comes in two flavors: sibling structures, i.e. nodes

with common parents, and co-parent structures, i.e. nodes with common children. In most Web and citation graph related application, the first assumption, namely that nodes with highly overlapping parent sets are likely to belong to the same category, seems to be more relevant (cf. [4]), but in general this will depend on the specific application.

One possible way of designing classifiers based on graph connectivity is to construct a kernel matrix based on pairwise links [11] and then to adopt a standard kernel method, e.g. Support Vector Machines (SVMs) [16] as a learning algorithm. However, a kernel matrix as the one proposed in [11] only represents *local* relationships among nodes, but completely ignores the *global* structure of the graph. The idea of exploiting global rather than local graph structure is widely used in other Web-related techniques, including Web page ranking [2, 13], finding similar Web pages [7], detecting Web communities [13, 9] and so on. The major innovation of this paper is a general regularization framework on directed graphs, in which the directionality and global relationships are considered, and a computationally attractive classification algorithm, which is derived from the proposed regularization framework.

## 2 Regularization Framework

### 2.1 Preliminaries

A directed graph $\Gamma = (V, E)$ consists of a set of vertices, denoted by $V$ and a set of edges, denoted by $E \subseteq V \times V$. Each edge is an ordered pair of nodes $[u, v]$ representing a directed connection from $u$ to $v$. We do not allow self loops, i.e. $[v, v] \notin E$ for all $v \in V$. In a weighted directed graph, a weight function $w : V \times V \to \mathbb{R}_+$ is associated with $\Gamma$, satisfying $w([u, v]) = 0$ if and only if $[u, v] \notin E$. Typically, we can equip a directed graph with a canonical weight function by defining $w([u, v]) \equiv 1$ if and only if $[u, v] \in E$. The in-degree $p(v)$ and out-degree $q(v)$ of a vertex $v \in V$, respectively, are defined as

$$p(v) \equiv \sum_{\{u | [u,v] \in E\}} w([u, v]), \quad \text{and} \quad q(v) \equiv \sum_{\{u | [v,u] \in E\}} w([v, u]). \tag{1}$$

Let $\mathcal{H}(V)$ denote the space of functions $f : V \to \mathbb{R}$, which assigns a real value $f(v)$ to each vertex $v$. The function $f$ can be represented as a column vector in $\mathbb{R}^{|V|}$, where $|V|$ denotes the number of the vertices in $V$. The function space $\mathcal{H}(V)$ can be endowed with the usual inner product:

$$\langle f, g \rangle = \sum_v f(v) g(v). \tag{2}$$

Accordingly, the norm of the function induced from the inner product is $\|f\| = \sqrt{\langle f, f \rangle}$.

### 2.2 Bipartite Graphs

A bipartite graph $G = (H, A, L)$ is a special type of directed graph that consists of two sets of vertices, denoted by $H$ and $A$ respectively, and a set of edges (or links), denoted by $L \subseteq H \times A$. In a bipartite graph, each edge connects a vertex in $H$ to a vertex in $A$. Any directed graph $\Gamma = (V, E)$ can be regarded as a bipartite graph using the following simple construction [14]: $H \equiv \{h | h \in V, q(h) > 0\}$, $A \equiv \{a | a \in V, p(a) > 0\}$, and $L \equiv E$. Figure 1 depicts the construction of the bipartite graph. Notice that vertices of the original graph $\Gamma$ may appear in both vertex sets $H$ and $A$ of the constructed bipartite graph.

The intuition behind the construction of the bipartite graph is provided by the so-called *hub* and *authority* web model introduced by Kleinberg [13]. The model distinguishes between two types of Web pages: *authoritative pages*, which are pages relevant to some topic, and *hub pages*, which are pages pointing to relevant pages. Note that some Web pages can

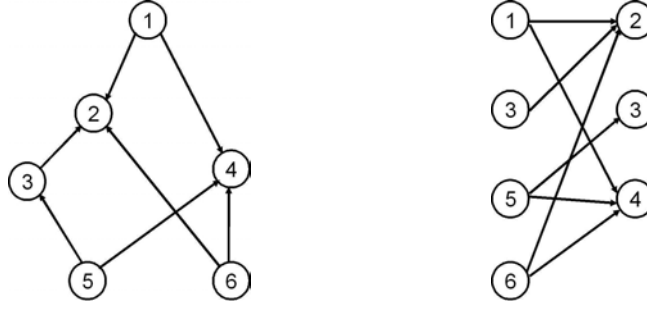

Figure 1: Constructing a bipartite graph from a directed one. Left: directed graph. Right: bipartite graph. The hub set $H = \{1, 3, 5, 6\}$, and the authority set $A = \{2, 3, 4\}$. Notice that the vertex indexed by 3 is simultaneously in the hub and authority set.

simultaneously be both hub and authority pages (see Figure 1). Hubs and authorities exhibit a *mutually reinforcement relationship*: a good hub node points to many good authorities and a good authority node is pointed to by many good hubs. It is interesting to note that in general there is no direct link from one authority to another. It is the hub pages that *glue together* authorities on a common topic.

According to Kleinberg's model, we suggestively call the vertex set $H$ in the bipartite graph the *hub set*, and the vertex set $A$ the *authority set*.

### 2.3 Smoothness Functionals

If two distinct vertices $u$ and $v$ in the authority set $A$ are co-linked by vertex $h$ in the hub set $H$ as shown in the left panel of Figure 2, then we think that $u$ and $v$ are likely to be related, and the co-linkage strength induced by $h$ between $u$ and $v$ can be measured by

$$c_h([u, v]) = \frac{w([h, u])w([h, v])}{q(h)}. \tag{3}$$

In addition, we define $c_h(v, v) = 0$ for all $v$ in the authority set $A$ and for all $h$ in the hub set $H$. Such a relevance measure can be naturally understood in the situation of citation networks. If two articles are simultaneously cited by some other article, then this should make it more likely that both articles deal with a similar topics. Moreover, the more articles cite both articles together, the more significant the connection. A natural question arising in this context is why the relevance measure is further normalized by out-degree. Let us consider the following two web sites: *Yahoo!* and *kernel machines*. General interest portals like Yahoo! consists of pages having a large number of diverse hyperlinks. The fact that two web pages are co-linked by Yahoo! does not establish a significant connection between them. In contrast, the pages on the kernel machine Web site have much fewer hyperlinks, but the Web pages pointed to are closely related in topic.

Let $f$ denote a function defined on the authority set $A$. The *smoothness* of function $f$ can be measured by the following functional:

$$\Omega_A(f) = \frac{1}{2} \sum_{u,v} \sum_h c_h([u, v]) \left( \frac{f(u)}{\sqrt{p(u)}} - \frac{f(v)}{\sqrt{p(v)}} \right)^2. \tag{4}$$

The smoothness functional penalizes large differences in function values for vertices in the authority set $A$ that are strongly related. Notice that the function values are normalized by

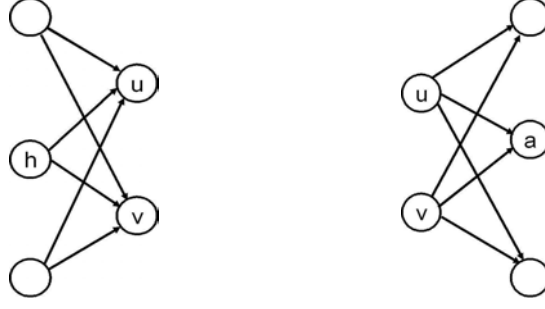

Figure 2: Link and relevance. Left panel: vertices $u$ and $v$ in the authority set $A$ are co-linked by vertex $h$ in the hub set $H$. Right panel: vertices $u$ and $v$ in the hub set $H$ co-link vertex $a$ in the authority set $A$.

in-degree. For the Web graph, the explanation is similar to the one given before. Many web pages contain links to popular sites like the Google search engine. This does not mean though that all these Web pages share a common topic. However, if two web pages point to web page like the one of the *Learning with Kernels* book, it is likely to express a common interest for kernel methods.

Now define a linear operator $T : \mathcal{H}(A) \to \mathcal{H}(H)$ by

$$(Tf)(h) = \sum_a \frac{w([h,a])}{\sqrt{q(h)p(a)}} f(a). \tag{5}$$

Then its adjoint $T^* : \mathcal{H}(H) \to \mathcal{H}(A)$ is given by

$$(T^*f)(a) = \sum_h \frac{w([h,a])}{\sqrt{q(h)p(a)}} f(h). \tag{6}$$

These two operators $T$ and $T^*$ were also implicitly suggested by [8] for developing a new Web page ranking algorithm. Further define the operator $S_A : \mathcal{H}(A) \to \mathcal{H}(A)$ by composing $T$ and $T^*$, i.e.

$$S_A = T^*T, \tag{7}$$

and the operator $\Delta_A : \mathcal{H}(A) \to \mathcal{H}(A)$ by

$$\Delta_A = I - S_A, \tag{8}$$

where $I$ denotes the identity operator. Then we can show the following (See Appendix A for the proof):

**Proposition 1.** $\Omega_A(f) = \langle f, \Delta_A f \rangle$.

Comparing with the combinatorial Laplace operator defined on undirected graphs [5], we can think of the operator $\Delta_A$ as a Laplacian but defined on the authority set of directed graphs. Note that Proposition 1 also shows that the Laplacian $\Delta_A$ is positive semi-definite. In fact, we can further show that the eigenvalues of the operator $S_A$ are scattered in $[0,1]$, and accordingly the eigenvalues of the Laplacian $\Delta_A$ fall into $[0,1]$.

Similarly, if two distinct vertices $u$ and $v$ co-link vertex $a$ in the authority set $A$ as shown in right panel of Figure 2, then $u$ and $v$ are also thought to be related. The co-linkage strength between $u$ and $v$ induced by $a$ can be measured by

$$c_a([u,v]) = \frac{w([u,a])w([v,a])}{p(a)} . \tag{9}$$

and the smoothness of function $f$ on the hub set $H$ can be measured by:

$$\Omega_H(f) = \frac{1}{2} \sum_{u,v} \sum_a c_a([u,v]) \left( \frac{f(u)}{\sqrt{q(u)}} - \frac{f(v)}{\sqrt{q(v)}} \right)^2. \tag{10}$$

As before, one can define the operators $S_H = TT^*$ and $\Delta_H = I - S_H$ leading to the corresponding statement:

**Proposition 2.** $\Omega_H(f) = \langle f, \Delta_H f \rangle$.

Convexly combining together the two smoothness functionals (4) and (10), we obtain a smoothness measure of function $f$ defined on the whole vertex set $V$:

$$\Omega_\gamma(f) = \gamma \Omega_A(f) + (1-\gamma)\Omega_H(f), \ 0 \le \gamma \le 1, \tag{11}$$

where the parameter $\gamma$ weighs the relative importance between $\Omega_A(f)$ and $\Omega_H(f)$. Extend the operator $T$ to $\mathcal{H}(V)$ by defining $(Tf)(v) = 0$ if $v$ is only in the authority set $A$ and not in the hub set $H$. Similarly extend $T^*$ by defining $(T^*f)(v) = 0$ if $v$ is only in the hub set $H$ and not in the authority set $A$. Then, if the remaining operators are extended correspondingly, one can define the operator $S_\gamma : \mathcal{H}(V) \rightarrow \mathcal{H}(V)$ by

$$S_\gamma = \gamma S_A + (1-\gamma)S_H, \tag{12}$$

and the Laplacian on directed graphs $\Delta_\gamma : \mathcal{H}(V) \rightarrow \mathcal{H}(V)$ by

$$\Delta_\gamma = I - S_\gamma. \tag{13}$$

Clearly, $\Delta_\gamma = \gamma \Delta_A + (1-\gamma)\Delta_H$. By Proposition 1 and 2, it is easy to see that:

**Proposition 3.** $\Omega_\gamma(f) = \langle f, \Delta_\gamma f \rangle$.

### 2.4 Regularization

Define a function $y$ in $\mathcal{H}(V)$ in which $y(v) = 1$ or $-1$ if vertex $v$ is labeled as positive or negative, and 0, if it is not labeled. The classification problem can be regarded as the problem of finding a function $f$, which reproduces the target function $y$ to a sufficient degree of accuracy while being smooth in a sense quantified by the above smoothness functional. A formalization of this idea leads to the following optimization problem:

$$f^* = \underset{f \in \mathcal{H}(V)}{\operatorname{argmin}} \left\{ \Omega_\gamma(f) + \frac{\mu}{2} \|f - y\|^2 \right\}. \tag{14}$$

The final classification of vertex $v$ is obtained as $\operatorname{sign} f^*(v)$. The first term in the bracket is called the *smoothness term* or *regularizer*, which measures the smoothness of function $f$, and the second term is called the *fitting term*, which measures its closeness to the given function $y$. The trade-off between these two competitive terms is captured by a positive parameter $\mu$. Successively smoother solutions $f^*$ can be obtained by decreasing $\mu \rightarrow 0$.

**Theorem 4.** *The solution $f^*$ of the optimization problem (14) satisfies*

$$\Delta_\gamma f^* + \mu(f^* - y) = 0.$$

*Proof.* By Proposition 3, we have

$$(\Delta_\gamma f)(v) = \left. \frac{\partial \Omega_\gamma(f)}{\partial f} \right|_v.$$

Differentiating the cost function in the bracket of (14) with respect to function $f$ completes the proof. $\square$

**Corollary 5.** *The solution $f^*$ of the optimization problem (14) is*

$$f^* = (1-\alpha)(I - \alpha S_\gamma)^{-1}y, \quad where \ \alpha = 1/(1+\mu).$$

It is worth noting that the closed form solution presented by Corollary 5 shares the same appearance as the algorithm proposed by [17], which operates on undirected graphs.

# 3 Experiments

We considered the Web page categorization task on the WebKB dataset [6]. We only addressed a subset which contains the pages from the four universities: Cornell, Texas, Washington, Wisconsin. We removed pages without incoming or outgoing links, resulting in 858, 825, 1195 and 1238 pages respectively, for a total of 4116. These pages were manually classified into the following seven categories: *student, faculty, staff, department, course, project* and *other*. We investigated two different classification tasks. The first is used to illustrate the significance of connectivity information in classification, whereas the second one stresses the importance of preserving the directionality of edges. We may assign a weight to each hyperlink according to the textual content of web pages or the anchor text contained in hyperlinks. However, here we are only interested in how much we can obtain from link structure only and hence adopt the canonical weight function defined in Section 2.1.

We first study an extreme classification problem: predicting which university the pages belong to from very few labeled training examples. Since pages within a university are well-linked, and cross links between different universities are rare, we can imagine that few training labels are enough to exactly classify pages based on link information only. For each of the universities, we in turn viewed corresponding pages as positive examples and the pages from the remaining universities as negative examples. We have randomly draw two pages as the training examples under the constraint that there is at least one labeled instance for each class. Parameters were set to $\gamma = 0.50$, and $\alpha = 0.95$. (In fact, in this experiment the tuning parameters have almost no influence on the result.) Since the Web graph is not connected, some small isolated subgraphs possibly do not contain labeled instances. The values of our classifying function on the pages contained in these subgraphs will be zeros and we simply think of these pages as negative examples. This is consistent with the search engine ranking techniques [2, 13]. We compare our method with SVMs using a kernel matrix $K$ constructed as $K = W^T W$ [11], where $W$ denotes the adjacency matrix of the web graph and $W^T$ denotes the transpose of $W$. The test errors averaged over 100 training sample sets for both our method and SVMs are summarized into the following table:

|  | Cornel | Texas | Washington | Wisconsin |
|---|---|---|---|---|
| our method | 0.03 ($\pm$ 0.00) | 0.02 ($\pm$ 0.01) | 0.01 ($\pm$ 0.00) | 0.02 ($\pm$ 0.00) |
| SVMs | 0.42 ($\pm$ 0.03) | 0.39 ($\pm$ 0.03) | 0.40 ($\pm$ 0.02) | 0.43 ($\pm$ 0.02) |

However, to be fair, we should state that the kernel matrix that we used in the SVM may not be the best possible kernel matrix for this task — this is an ongoing research issue which is not the topic of the present paper.

The other investigated task is to discriminate the student pages in a university from the non-student pages in the same university. As a baseline we have applied our regularization method on the undirected graph [17] obtained by treating links as undirected or bidirectional, i.e., the affinity matrix is defined to be $W^T + W$. We use the AUC scores to measure the performances of the algorithms. The experimental results in Figure 3(a)-3(d) clearly demonstrate that taking the directionality of edges into account can yield substantial accuracy gains. In addition, we also studied the influence of different choices for the parameters $\gamma$ and $\alpha$; we used the Cornell Web for that purpose and sampled 10 labeled training pages. Figure 3(e) show that relatively small values of $\alpha$ are more suitable. We think that is because the subgraphs in each university are quite small, limiting the information conveyed in the graph structure. The influence of $\gamma$ is shown in Figure 3(f). The performance curve shows that large values for $\gamma$ are preferable. This confirms the conjecture that co-link structure among authority pages is much more important than within the hub set.

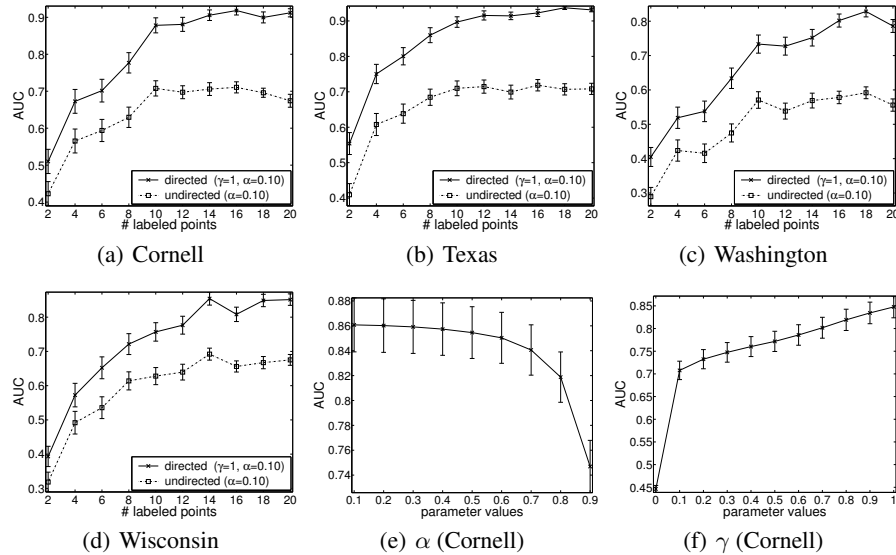

Figure 3: Classification on the WebKB dataset. Figure (a)-(d) depict the AUC scores of the directed and undirected regularization methods on the classification problem *student vs. non-student* in each university. Figure (e)-(f) illustrate the influences of the different choices of the parameters $\alpha$ and $\gamma$.

## 4   Conclusions

We proposed a general regularization framework on directed graphs, which has been validated on a real-word Web data set. The remaining problem is how to choose the suitable parameters contained in this approach. In addition, it is worth noticing that this framework can be applied without any essential changes to bipartite graphs, e.g. to graphs describing customers' purchase behavior in market basket analysis. Moreover, in the absence of labeled instances, this framework can be utilized in an unsupervised setting as a (spectral) clustering method for directed or bipartite graphs. Due to lack of space, we have not been able to give a thorough discussion of these topics.

**Acknowledgments**   We would like to thank David Gondek for his help on this work.

## References

[1] M. Belkin, I. Matveeva, and P. Niyogi. Regularization and regression on large graphs. In *COLT*, 2004.

[2] S. Brin and L. Page. The anatomy of a large scale hypertextual web search engine. In *Proc. 7th Intl. WWW Conf.*, 1998.

[3] A. Broder, R. Kumar, F. Maghoul, P. Raghavan, S. Rajagopalan, R. Stata, A. Tomkins, and J. Wiener. Graph structure in the Web. In *Proc. 9th Intl. WWW Conf.*, 2000.

[4] S. Chakrabarti, B. Dom, and P. Indyk. Enhanced hypertext categorization using hyperlinks. In *Proc. ACM SIGMOD Conf.*, 1998.

[5] F. Chung. *Spectral Graph Theory*. Number 92 in Regional Conference Series in Mathematics. American Mathematical Society, 1997.

[6] M. Craven, D. DiPasquo, D. Freitag, A. McCallum, T. Mitchell, K. Nigam, and S. Slattery. Learning to extract symbolic knowledge from the World Wide Web. In *Proc. 15th National Conf. on Artificial Intelligence*, 1998.

[7] J. Dean and M. Henzinger. Finding related Web pages in the World Wide Web. In *Proc. 8th Intl. WWW Conf.*, 1999.

[8] C. Ding, X. He, P. Husbands, H. Zha, and H. D. Simon. PageRank, HITS and a unified framework for link analysis. In *Proc. 25th ACM SIGIR Conf.*, 2001.

[9] G. Flake, S. Lawrence, C. L. Giles, and F. Coetzee. Self-organization and identification of Web communities. *IEEE Computer*, 35(3):66–71, 2002.

[10] C. Lee Giles, K. Bollacker, and S. Lawrence. CiteSeer: An automatic citation indexing system. In *Proc. 3rd ACM Conf. on Digital Libraries*, 1998.

[11] T. Joachims, N. Cristianini, and J. Shawe-Taylor. Composite kernels for hypertext categorisation. In *ICML*, 2001.

[12] T. Joachims. Transductive learning via spectral graph partitioning. In *ICML*, 2003.

[13] J. Kleinberg. Authoritative sources in a hyperlinked environment. *Journal of the ACM*, 46(5):604–632, 1999.

[14] R. Lempel and S. Moran. SALSA: the stochastic approach for link-structure analysis. *ACM Transactions on Information Systems*, 19(2):131–160, 2001.

[15] A. Smola and R. Kondor. Kernels and regularization on graphs. In *Learning Theory and Kernel Machines*. Springer-Verlag, Berlin-Heidelberg, 2003.

[16] V. N. Vapnik. *Statistical learning theory*. Wiley, NY, 1998.

[17] D. Zhou, O. Bousquet, T. N. Lal, J. Weston, and B. Schölkopf. Learning with local and global consistency. In *NIPS*, 2003.

[18] X. Zhu, Z. Ghahramani, and J. Lafferty. Semi-supervised learning using Gaussian fields and harmonic functions. In *ICML*, 2003.

## A  Proof of Proposition 1

Expand the right site of Equ. (4):

$$
\begin{aligned}
\Omega_A(f) &= \sum_{u,v} \sum_h c_h([u,v]) \left( \frac{f^2(u)}{p(u)} - \frac{f(u)f(v)}{\sqrt{p(u)p(v)}} \right) \\
&= \sum_u \left( \sum_v \sum_h c_h([u,v]) \right) \frac{f^2(u)}{p(u)} - \sum_{u,v} \sum_h c_h([u,v]) \frac{f(u)f(v)}{\sqrt{p(u)p(v)}}.
\end{aligned}
\tag{15}
$$

By substituting Equ. (3), the first term in the above equality can be rewritten as

$$
\begin{aligned}
& \sum_u \left( \sum_v \sum_h \frac{w([h,u])w([h,v])}{q(h)} \right) \frac{f^2(u)}{p(u)} \\
&= \sum_u \left( \sum_h \frac{w([h,u])}{p(u)} \right) f^2(u) = \sum_u f^2(u).
\end{aligned}
\tag{16}
$$

In addition, the second term in Equ. (15) can be transformed into

$$
\begin{aligned}
& \sum_{u,v} \sum_h \frac{w([h,u])w([h,v])}{q(h)} \frac{f(u)f(v)}{\sqrt{p(u)p(v)}} \\
&= \sum_{u,v} \sum_h f(u) \frac{w([h,u])}{\sqrt{q(h)p(u)}} \frac{w([h,v])}{\sqrt{q(h)p(v)}} f(v).
\end{aligned}
\tag{17}
$$

Substituting Equ. (16) and (17) into (15), we have

$$
\Omega_A(f) = \sum_u f^2(u) - \sum_{u,v} \sum_h f(u) \frac{w([h,u])}{\sqrt{q(h)p(u)}} \frac{w([h,v])}{\sqrt{q(h)p(v)}} f(v).
\tag{18}
$$

This completes the proof.